# Bundle Methods for Machine Learning

**Alexander J. Smola, S.V. N. Vishwanathan, Quoc V. Le**
NICTA and Australian National University, Canberra, Australia
*Alex.Smola@gmail.com, {SVN.Vishwanathan, Quoc.Le}@nicta.com.au*

## Abstract

We present a globally convergent method for regularized risk minimization problems. Our method applies to Support Vector estimation, regression, Gaussian Processes, and any other regularized risk minimization setting which leads to a convex optimization problem. SVMPerf can be shown to be a special case of our approach. In addition to the unified framework we present tight convergence bounds, which show that our algorithm converges in $O(1/\epsilon)$ steps to $\epsilon$ precision for general convex problems and in $O(\log(1/\epsilon))$ steps for continuously differentiable problems. We demonstrate in experiments the performance of our approach.

## 1 Introduction

In recent years optimization methods for convex models have seen significant progress. Starting from the active set methods described by Vapnik [17] increasingly sophisticated algorithms for solving regularized risk minimization problems have been developed. Some of the most exciting recent developments are SVMPerf [5] and the Pegasos gradient descent solver [12]. The former computes gradients of the current solution at every step and adds those to the optimization problem. Joachims [5] prove an $O(1/\epsilon^2)$ rate of convergence. For Pegasos Shalev-Shwartz et al. [12] prove an $O(1/\epsilon)$ rate of convergence, which suggests that Pegasos should be much more suitable for optimization.

In this paper we extend the ideas of SVMPerf to general convex optimization problems and a much wider class of regularizers. In addition to this, we present a formulation which does not require the solution of a quadratic program whilst in practice enjoying the same rate of convergence as algorithms of the SVMPerf family. Our error analysis shows that the rates achieved by this algorithm are considerably better than what was previously known for SVMPerf, namely the algorithm enjoys $O(1/\epsilon)$ convergence and $O(\log(1/\epsilon))$ convergence, whenever the loss is sufficiently smooth. An important feature of our algorithm is that it *automatically* takes advantage of smoothness in the problem.

Our work builds on [15], which describes the basic extension of SVMPerf to general convex problems. The current paper provides a) significantly improved performance bounds which match better what can be observed in practice and which apply to a wide range of regularization terms, b) a variant of the algorithm which does *not* require quadratic programming, yet enjoys the same fast rates of convergence, and c) experimental data comparing the speed of our solver to Pegasos and SVMPerf. Due to space constraints we relegate the proofs to an technical report [13].

## 2 Problem Setting

Denote by $x \in \mathcal{X}$ and $y \in \mathcal{Y}$ patterns and labels respectively and let $l(x, y, w)$ be a loss function which is convex in $w \in \mathcal{W}$, where either $\mathcal{W} = \mathbb{R}^d$ (linear classifier), or $\mathcal{W}$ is a Reproducing Kernel Hilbert Space for kernel methods. Given a set of $m$ training patterns $\{x_i, y_i\}_{i=1}^m$ the regularized risk

functional which many estimation methods strive to minimize can be written as

$$J(w) := R_{\text{emp}}(w) + \lambda\Omega(w) \text{ where } R_{\text{emp}}(w) := \frac{1}{m}\sum_{i=1}^{m} l(x_i, y_i, w). \tag{1}$$

$\Omega(w)$ is a smooth convex regularizer such as $\frac{1}{2}\|w\|^2$, and $\lambda > 0$ is a regularization term. Typically $\Omega$ is cheap to compute and to minimize whereas the empirical risk term $R_{\text{emp}}(w)$ is computationally expensive to deal with. For instance, in the case of intractable graphical models it requires approximate inference methods such as sampling or semidefinite programming. To make matters worse the number of training observations $m$ may be huge. We assume that the empirical risk $R_{\text{emp}}(w)$ is *nonnegative*.

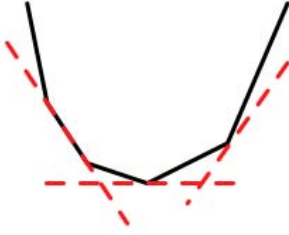

**Figure 1:** A lower bound on the convex empirical risk $R_{\text{emp}}(w)$ obtained by computing three tangents on the entire function.

If $J$ is differentiable we can use standard quasi-Newtons methods like LBFGS even for large values of $m$ [8]. Unfortunately, it is not straightforward to extend these algorithms to optimize a non-smooth objective. In such cases one has to resort to bundle methods [3], which are based on the following elementary observation: for convex functions a first order Taylor approximation is a lower bound. So is the maximum over a set of Taylor approximations. Furthermore, the Taylor approximation is exact at the point of expansion. The idea is to replace $R_{\text{emp}}[w]$ by these lower bounds and to optimize the latter in conjunction with $\Omega(w)$. Figure 1 gives geometric intuition. In the remainder of the paper we will show that 1) This extends a number of existing algorithms; 2) This method enjoys good rates of convergence; and 3) It works well in practice.

Note that there is no need for $R_{\text{emp}}[w]$ to decompose into individual losses in an additive fashion. For instance, scores, such as Precision@k [4], or SVM Ranking scores do not satisfy this property. Likewise, estimation problems which allow for an unregularized common constant offset or adaptive margin settings using the $\nu$-trick fall into this category. The only difference is that in those cases the derivative of $R_{\text{emp}}[w]$ with respect to $w$ no more decomposes trivially into a sum of gradients.

## 3 Bundle Methods

### 3.1 Subdifferential and Subgradient

Before we describe the bundle method, it is necessary to clarify a key technical point. The subgradient is a generalization of gradients appropriate for convex functions, including those which are not necessarily smooth. Suppose $w$ is a point where a convex function $F$ is finite. Then a subgradient is the normal vector of any tangential supporting hyperplane of $F$ at $w$. Formally $\mu$ is called a subgradient of $F$ at $w$ if, and only if,

$$F(w') \geq F(w) + \langle w' - w, \mu \rangle \quad \forall w'. \tag{2}$$

The set of all subgradients at a point is called the subdifferential, and is denoted by $\partial_w F(w)$. If this set is not empty then $F$ is said to be *subdifferentiable at* $w$. On the other hand, if this set is a singleton then, the function is said to be *differentiable* at $w$.

### 3.2 The Algorithm

Denote by $w_t \in \mathcal{W}$ the values of $w$ which are obtained by successive steps of our method, Let $a_t \in \mathcal{W}$, $b_t \in \mathbb{R}$, and set $w_0 = 0, a_0 = 0, b_0 = 0$. Then, the Taylor expansion coefficients of $R_{\text{emp}}[w_t]$ can be written as

$$a_{t+1} := \partial_w R_{\text{emp}}(w_t) \text{ and } b_{t+1} := R_{\text{emp}}(w_t) - \langle a_{t+1}, w_t \rangle. \tag{3}$$

Note that we do not require $R_{\text{emp}}$ to be differentiable: if $R_{\text{emp}}$ is not differentiable at $w_t$ we simply choose *any* element of the subdifferential as $a_{t+1}$. Since each Taylor approximation is a lower bound, we may take their maximum to obtain that $R_{\text{emp}}(w) \geq \max_t \langle a_t, w \rangle + b_t$. Moreover, by

**Algorithm 1** Bundle Method($\epsilon$)

---

Initialize $t = 0, w_0 = 0, a_0 = 0, b_0 = 0$, and $J_0(w) = \lambda\Omega(w)$
**repeat**
    Find minimizer $w_t := \operatorname{argmin}_w J_t(w)$
    Compute gradient $a_{t+1}$ and offset $b_{t+1}$.
    Increment $t \leftarrow t + 1$.
**until** $\epsilon_t \le \epsilon$

---

virtue of the fact that $R_{\mathrm{emp}}$ is a non-negative function we can write the following lower bounds on $R_{\mathrm{emp}}$ and $J$ respectively:

$$R_t(w) := \max_{t' \le t} \langle a_{t'}, w \rangle + b_{t'} \text{ and } J_t(w) := \lambda\Omega(w) + R_t(w). \tag{4}$$

By construction $R_{t'} \le R_t \le R_{\mathrm{emp}}$ and $J_{t'} \le J_t \le J$ for all $t' \le t$. Define

$$w^* := \operatorname*{argmin}_w J(w), \qquad\qquad \gamma_t := J_{t+1}(w_t) - J_t(w_t),$$

$$w_t := \operatorname*{argmin}_w J_t(w), \qquad \text{and} \qquad \epsilon_t := \min_{t' \le t} J_{t'+1}(w_{t'}) - J_t(w_t).$$

The following lemma establishes some useful properties of $\gamma_t$ and $\epsilon_t$.

**Lemma 1** *We have $J_{t'}(w_{t'}) \le J_t(w_t) \le J(w^*) \le J(w_t) = J_{t+1}(w_t)$ for all $t' \le t$. Furthermore, $\epsilon_t$ is monotonically decreasing with $\epsilon_t - \epsilon_{t+1} \ge J_{t+1}(w_{t+1}) - J_t(w_t) \ge 0$. Also, $\epsilon_t$ upper bounds the distance from optimality via $\gamma_t \ge \epsilon_t \ge \min_{t' \le t} J(w_{t'}) - J(w^*)$.*

### 3.3 Dual Problem

Optimization is often considerably easier in the dual space. In fact, we will show that we need not know $\Omega(w)$ at all, instead it is sufficient to work with its Fenchel-Legendre dual $\Omega^*(\mu) := \sup_w \langle w, \mu \rangle - \Omega(w)$. If $\Omega^*$ is a so-called Legendre function [e.g. 10] the $w$ at which the supremum is attained can be written as $w = \partial_\mu \Omega^*(\mu)$. In the sequel we will always assume that $\Omega^*$ is twice differentiable and Legendre. Examples include $\Omega^*(\mu) = \frac{1}{2}\|\mu\|^2$ or $\Omega^*(\mu) = \sum_i \exp[\mu]_i$.

**Theorem 2** *Let $\alpha \in \mathbb{R}^t$, denote by $A = [a_1, \ldots, a_t]$ the matrix whose columns are the (sub)gradients, let $b = [b_1, \ldots, b_t]$. The dual problem of*

$$\operatorname*{minimize}_w J_t(w) := \max_{t' \le t} \langle a_{t'}, w \rangle + b_{t'} + \lambda\Omega(w) \text{ is} \tag{5}$$

$$\operatorname*{maximize}_\alpha J_t^*(\alpha) := -\lambda\Omega^*(-\lambda^{-1}A\alpha) + \alpha^\top b \text{ subject to } \alpha \ge 0 \text{ and } \|\alpha\|_1 = 1. \tag{6}$$

*Furthermore, the optimal $w_t$ and $\alpha_t$ are related by the dual connection $w_t = \partial\Omega^*(-\lambda^{-1}A\alpha_t)$.*

Recall that for $\Omega(w) = \frac{1}{2}\|w\|_2^2$ the Fenchel-Legendre dual is given by $\Omega^*(\mu) = \frac{1}{2}\|\mu\|_2^2$. This is commonly used in SVMs and Gaussian Processes. The following corollary is immediate:

**Corollary 3** *Define $Q := A^\top A$, i.e. $Q_{uv} := \langle a_u, a_v \rangle$. For quadratic regularization, i.e. $\operatorname{minimize}_w \max(0, \max_{t' \le t} \langle a_{t'}, w \rangle + b_{t'}) + \frac{\lambda}{2}\|w\|_2^2$ the dual becomes*

$$\operatorname*{maximize}_\alpha -\frac{1}{2\lambda}\alpha^\top Q\alpha + \alpha^\top b \text{ subject to } \alpha \ge 0 \text{ and } \|\alpha\|_1 = 1. \tag{7}$$

This means that for quadratic regularization the dual optimization problem is a *quadratic* program where the number of variables equals the number of gradients computed previously. Since $t$ is typically in the order of 10s to 100s, the resulting QP is very cheap to solve. In fact, we don't even need to know the gradients explicitly. All that is required to define the QP are the inner products between gradient vectors $\langle a_u, a_v \rangle$. Later in this section we propose a variant which does away with the quadratic program altogether while preserving most of the appealing convergence properties of Algorithm 1.

## 3.4 Examples

**Structured Estimation**   Many estimation problems [14, 16] can be written in terms of a piecewise linear loss function

$$l(x, y, w) = \max_{y' \in \mathcal{Y}} \langle \phi(x, y') - \phi(x, y), w \rangle + \Delta(y, y') \tag{8}$$

for some suitable joint feature map $\phi$, and a loss function $\Delta(y, y')$. It follows from Section 3.1 that a subdifferential of (8) is given by

$$\partial_w l(x, y, w) = \phi(x, y^*) - \phi(x, y) \text{ where } y^* := \underset{y' \in \mathcal{Y}}{\operatorname{argmax}} \langle \phi(x, y') - \phi(x, y), w \rangle + \Delta(y, y'). \tag{9}$$

Since $R_{\text{emp}}$ is defined as a summation of loss terms, this allows us to apply Algorithm 1 directly for risk minimization: at every iteration $t$ we find all maximal constraint violators for each $(x_i, y_i)$ pair and compute the composite gradient vector. This vector is then added to the convex program we have so far.

Joachims [5] pointed out this idea for the special case of $\phi(x, y) = y\phi(x)$ and $y \in \{\pm 1\}$, that is, binary loss. Effectively, by defining a joint feature map as the sum over individual feature maps and by defining a joint loss $\Delta$ as the sum over individual losses SVMPerf performs exactly the same operations as we described above. Hence, for losses of type (8) our algorithm is a direct extension of SVMPerf to structured estimation.

**Exponential Families**   One of the advantages of our setting is that it applies to *any* convex loss function, as long as there is an efficient way of computing the gradient. That is, we can use it for cases where we are interested in modeling

$$p(y|x; w) = \exp(\langle \phi(x, y), w \rangle - g(w|x)) \text{ where } g(w|x) = \log \int_{\mathcal{Y}} \exp \langle \phi(x, y'), w \rangle \, dy' \tag{10}$$

That is, $g(w|x)$ is the conditional log-partition function. This type of losses includes settings such as Gaussian Process classification and Conditional Random Fields [1]. Such settings have been studied by Lee et al. [6] in conjunction with an $\ell_1$ regularizer $\Omega(w) = \|w\|_1$ for structure discovery in graphical models. Choosing $l$ to be the negative log-likelihood it follows that

$$R_{\text{emp}}(w) = \sum_{i=1}^{m} g(w|x_i) - \langle \phi(x_i, y_i), w \rangle \text{ and } \partial_w R_{\text{emp}}(w) = \sum_{i=1}^{m} \mathbf{E}_{y' \sim p(y'|x_i; w)} \left[ \phi(x_i, y') \right] - \phi(x_i, y_i).$$

This means that column generation methods are therefore directly applicable to Gaussian Process estimation, a problem where large scale solvers were somewhat more difficult to find. It also shows that adding a new model becomes a matter of defining a new loss function and its corresponding gradient, rather than having to build a full solver from scratch.

## 4   Convergence Analysis

While Algorithm 1 is intuitively plausible, it remains to be shown that it has good rates of convergence. In fact, past results, such as those by Tsochantaridis et al. [16] suggest an $O(1/\epsilon^2)$ rate, which would make the application infeasible in practice.

We use a duality argument similar to those put forward in [11, 16], both of which share key techniques with [18]. The crux of our proof argument lies in showing that $\epsilon_t - \epsilon_{t+1} \geq J_{t+1}(w_{t+1}) - J_t(w_t)$ (see Theorem 4) is sufficiently bounded away from 0. In other words, since $\epsilon_t$ bounds the distance from the optimality, at every step Algorithm 1 makes sufficient progress towards the optimum. Towards this end, we first observe that by strong duality the values of the primal and dual problems (5) and (6) are equal at optimality. Hence, any progress in $J_{t+1}$ can be computed in the dual.

Next, we observe that the solution of the dual problem (6) at iteration $t$, denoted by $\alpha_t$, forms a feasible set of parameters for the dual problem (6) at iteration $t+1$ by means of the parameterization $(\alpha_t, 0)$, i.e. by padding $\alpha_t$ with a 0. The value of the objective function in this case equals $J_t(w_t)$.

To obtain a *lower* bound on the improvement due to $J_{t+1}(w_{t+1})$ we perform a line search along $((1-\eta)\alpha_t, \eta)$ in (6). The constraint $\eta \in [0, 1]$ ensures dual feasibility. We will bound this improvement in terms of $\gamma_t$. Note that, in general, solving the dual problem (6) results in an increase which is larger than that obtained via the line search. The line search is employed in the analysis only for analytic tractability. We aim to lower-bound $\epsilon_t - \epsilon_{t+1}$ in terms of $\epsilon_t$ and solve the resultant difference equation.

Depending on $J(w)$ we will be able to prove two different convergence results.

(a) For regularizers $\Omega(w)$ for which $\left\|\partial_\mu^2 \Omega^*(\mu)\right\| \leq H^*$ we first experience a regime of progress linear in $\gamma_t$ and a subsequent slowdown to improvements which are quadratic in $\gamma_t$.
(b) Under the above conditions, if furthermore $\left\|\partial_w^2 J(w)\right\| \leq H$, i.e. the Hessian of $J$ is bounded, we have linear convergence throughout.

We first derive lower bounds on the improvement $J_{t+1}(w_{t+1}) - J_t(w_t)$, then the fact that for (b) the bounds are better. Finally we prove the convergence rates by solving the difference equation in $\epsilon_t$. This reasoning leads to the following theorem:

**Theorem 4** *Assume that $\|\partial_w R_{\mathrm{emp}}(w)\| \leq G$ for all $w \in W$, where $W$ is some domain of interest containing all $w_{t'}$ for $t' \leq t$. Also assume that $\Omega^*$ has bounded curvature, i.e. let $\left\|\partial_\mu^2 \Omega^*(\mu)\right\| \leq H^*$ for all $\mu \in \left\{-\lambda^{-1}\bar{A}\bar{\alpha} \text{ where } \bar{\alpha} \geq 0 \text{ and } \|\bar{\alpha}\|_1 \leq 1\right\}$. In this case we have*

$$\epsilon_t - \epsilon_{t+1} \geq \tfrac{\gamma_t}{2}\min(1, \lambda\gamma_t/4G^2 H^*) \geq \tfrac{\epsilon_t}{2}\min(1, \lambda\epsilon_t/4G^2 H^*). \tag{11}$$

*Furthermore, if $\left\|\partial_w^2 J(w)\right\| \leq H$, then we have*

$$\epsilon_t - \epsilon_{t+1} \geq \begin{cases} \gamma_t/2 & \text{if } \gamma_t > 4G^2 H^*/\lambda \\ \lambda/8H^* & \text{if } 4G^2 H^*/\lambda \geq \gamma_t \geq H/2 \\ \lambda\gamma_t/4HH^* & \text{otherwise} \end{cases} \tag{12}$$

Note that the error keeps on halving initially and settles for a somewhat slower rate of convergence after that, whenever the Hessian of the overall risk is bounded from above. The reason for the difference in the convergence bound for differentiable and non-differentiable losses is that in the former case the gradient of the risk converges to 0 as we approach optimality, whereas in the former case, no such guarantees hold (e.g. when minimizing $|x|$ the (sub)gradient does not vanish at the optimum).

Two facts are worthwhile noting: a) The dual of many regularizers, e.g. squares norm, squared $\ell_p$ norm, and the entropic regularizer have bounded second derivative. See e.g. [11] for a discussion and details. Thus our condition $\left\|\partial_\mu^2 \Omega^*(\mu)\right\| \leq H^*$ is not unreasonable. b) Since the improvements decrease with the size of $\gamma_t$ we may replace $\gamma_t$ by $\epsilon_t$ in both bounds and conditions without any ill effect (the bound only gets worse). Applying the previous result we obtain a convergence theorem for bundle methods.

**Theorem 5** *Assume that $J(w) \geq 0$ for all $w$. Under the assumptions of Theorem 4 we can give the following convergence guarantee for Algorithm 1. For any $\epsilon < 4G^2 H^*/\lambda$ the algorithm converges to the desired precision after*

$$n \leq \log_2 \frac{\lambda J(0)}{G^2 H^*} + \frac{8G^2 H^*}{\lambda\epsilon} - 4 \tag{13}$$

*steps. If furthermore the Hessian of $J(w)$ is bounded, convergence to any $\epsilon \leq H/2$ takes at most the following number of steps:*

$$n \leq \log_2 \frac{\lambda J(0)}{4G^2 H^*} + \frac{4H^*}{\lambda}\max\left[0, H - 8G^2 H^*/\lambda\right] + \frac{4HH^*}{\lambda}\log(H/2\epsilon) \tag{14}$$

Several observations are in order: firstly, note that the number of iterations only depends *logarithmically* on how far the initial value $J(0)$ is away from the optimal solution. Compare this to the result of Tsochantaridis et al. [16], where the number of iterations is linear in $J(0)$.

Secondly, we have an $O(1/\epsilon)$ dependence in the number of iterations in the non-differentiable case. This matches the rate of Shalev-Shwartz et al. [12]. In addition to that, the convergence is $O(\log(1/\epsilon))$ for continuously differentiable problems.

Note that whenever $R_{\text{emp}}(w)$ is the average over many piecewise linear functions $R_{\text{emp}}(w)$ behaves essentially like a function with bounded Hessian as long as we are taking large enough steps not to "notice" the fact that the term is actually nonsmooth.

**Remark 6** *For $\Omega(w) = \frac{1}{2}\|w\|^2$ the dual Hessian is exactly $H^* = 1$. Moreover we know that $H \geq \lambda$ since $\left\|\partial_w^2 J(w)\right\| = \lambda + \left\|\partial_w^2 R_{\text{emp}}(w)\right\|$.*

Effectively the rate of convergence of the algorithm is governed by upper bounds on the primal and dual curvature of the objective function. This acts like a condition number of the problem — for $\Omega(w) = \frac{1}{2}w^\top Q w$ the dual is $\Omega^*(z) = \frac{1}{2}z^\top Q^{-1}z$, hence the largest eigenvalues of $Q$ and $Q^{-1}$ would have a significant influence on the convergence.

In terms of $\lambda$ the number of iterations needed for convergence is $O(\lambda^{-1})$. In practice the iteration count *does* increase with $\lambda$, albeit not as badly as predicted. This is likely due to the fact that the empirical risk $R_{\text{emp}}(w)$ is typically rather smooth and has a certain inherent curvature which acts as a natural regularizer in addition to the regularization afforded by $\lambda\Omega[w]$.

## 5  A Linesearch Variant

The convergence analysis in Theorem 4 relied on a one-dimensional line search. Algorithm 1, however, uses a more complex quadratic program to solve the problem. Since even the simple updates promise good rates of convergence it is tempting to replace the corresponding step in the bundle update. This can lead to considerable savings in particular for smaller problems, where the time spent in the quadratic programming solver is a substantial fraction of the total runtime.

To keep matters simple, we only consider quadratic regularization $\Omega(w) := \frac{1}{2}\|w\|^2$. Note that $J_{t+1}(\eta) := J_{t+1}^*((1-\eta)\alpha^t, \eta)$ is a quadratic function in $\eta$, regardless of the choice of $R_{\text{emp}}[w]$. Hence a line search only needs to determine first and second derivative as done in the proof of Theorem 4. It can be shown that $\partial_\eta J_{t+1}(0) = \gamma_t$ and $\partial_\eta^2 J_{t+1}(0) = -\frac{1}{\lambda}\left\|\partial_w J(w_t)\right\|^2 = -\frac{1}{\lambda}\left\|\lambda w_t + a_{t+1}\right\|^2$. Hence the optimal value of $\eta$ is given by

$$\eta = \min(1, \lambda\gamma_t / \|\lambda w_t + a_{t+1}\|_2^2). \tag{15}$$

This means that we may update $w_{t+1} = (1-\eta)w_t - \frac{\eta}{\lambda}a_{t+1}$. In other words, we need not store past gradients for the update. To obtain $\gamma_t$ note that we are computing $R_{\text{emp}}(w_t)$ as part of the Taylor approximation step. Finally, $R_t(w_t)$ is given by $\left[w^\top A + b\right]\alpha^t$, hence it satisfies the same update relations. In particular, the fact that $w^\top A\alpha = \lambda\|w\|^2$ means that the only quantity we need to cache is $b^\top \alpha^t$ as an auxiliary variable $r_t$ in order to compute $\gamma_t$ efficiently. Experiments show that this simplified algorithm has essentially the same convergence properties.

## 6  Experiments

In this section we show experimental results that demonstrate the merits of our algorithm and its analysis. Due to space constraints, we report results of experiments with two large datasets namely Astro-Physics (astro-ph) and Reuters-CCAT (reuters-ccat) [5, 12]. For a fair comparison with existing solvers we use the quadratic regularizer $\Omega(w) := \frac{\lambda}{2}\|w\|^2$, and the binary hinge loss.

In our first experiment, we address the rate of convergence and its dependence on the value of $\lambda$. In Figure 2 we plot $\epsilon_t$ as a function of iterations for various values of $\lambda$ using the QP solver at every iteration to solve the dual problem (6) to optimality. Initially, we observe super-linear convergence; this is consistent with our analysis. Surprisingly, even though theory predicts sub-linear speed of convergence for non-differentiable losses like the binary hinge loss (see (11)), our solver exhibits linear rates of convergence predicted only for differentiable functions (see (12)). We conjecture that the average over many piecewise linear functions, $R_{\text{emp}}(w)$, behaves essentially like a smooth function. As predicted, the convergence speed is inversely proportional to the value of $\lambda$.

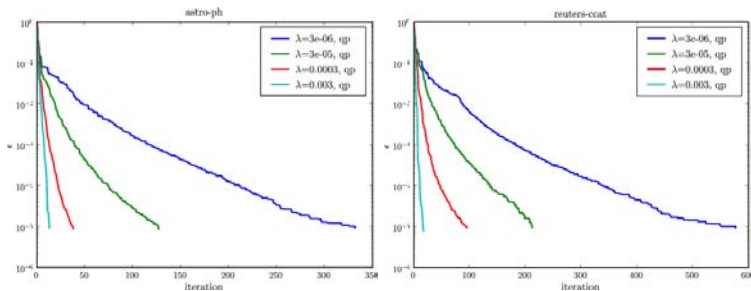

Figure 2: We plot $\epsilon_t$ as a function of the number of iterations. Note the logarithmic scale in $\epsilon_t$. Left: astro-ph; Right: reuters-ccat.

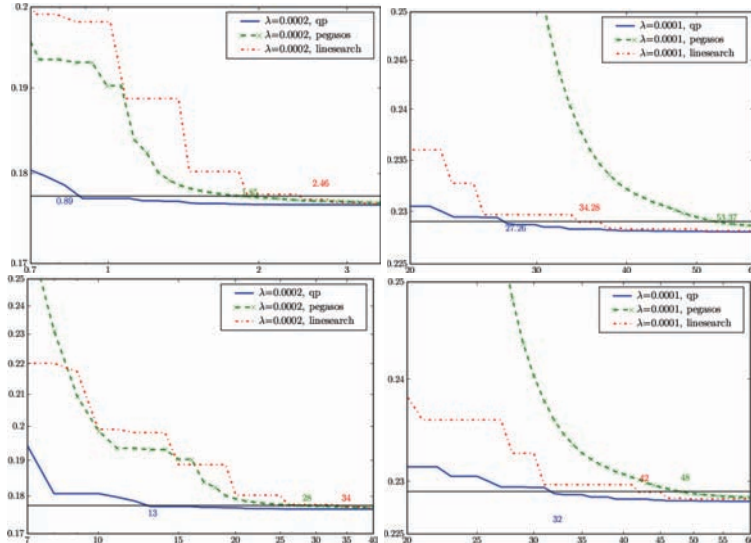

Figure 3: Top: Objective function value as a function of time. Bottom: Objective function value as a function of iterations. Left: astro-ph, Right: reuters-ccat. The black line indicates the final value of the objective function + 0.001 .

In our second experiment, we compare the convergence speed of two variants of the bundle method, namely, with a QP solver in the inner loop (which essentially boils down to SVMPerf) and the line search variant which we described in Section 5. We contrast these solvers with Pegasos [12] in the *batch* setting. Following [5] we set $\phi = 10^{-4}$ for reuters-ccat and $\phi = 2 \, 10^{-4}$ for astro-ph.

Figure 3 depicts the evolution of the primal objective function value as a function of both CPU time as well as the number of iterations. Following Shalev-Shwartz et al. [12] we investigate the time required by various solvers to reduce the objective value to within $0\,001$ of the optimum. This is depicted as a black horizontal line in our plots. As can be seen, Pegasos converges to this region quickly. Nevertheless, both variants of the bundle method converge to this value even faster (line search is slightly slower than Pegasos on astro-ph, but this is not always the case for many other large datasets we tested on). Note that both line search and Pegasos converge to within 0.001 precision rather quickly, but they require a large number of iterations to converge to the optimum.

# 7   Related Research

Our work is closely related to Shalev-Shwartz and Singer [11] who prove mistake bounds for online algorithms by lower bounding the progress in the dual. Although not stated explicitly, essentially the same technique of lower bounding the dual improvement was used by Tsochantaridis et al. [16] to show polynomial time convergence of the SVMStruct algorithm. The main difference however is that Tsochantaridis et al. [16] only work with a quadratic objective function while the framework

proposed by Shalev-Shwartz and Singer [11] can handle arbitrary convex functions. In both cases, a weaker analysis led to $O(1/\epsilon^2)$ rates of convergence for nonsmooth loss functions. On the other hand, our results establish a $O(1/\epsilon)$ rate for nonsmooth loss functions and $O(\log(1/\epsilon))$ rates for smooth loss functions under mild technical assumptions.

Another related work is SVMPerf [5] which solves the SVM estimation problem in linear time. SVMPerf finds a solution with accuracy $\epsilon$ in $O(md/(\lambda\epsilon^2))$ time, where the $m$ training patterns $x_i \in \mathbb{R}^d$. This bound was improved by Shalev-Shwartz et al. [12] to $\tilde{O}(1/\lambda\delta\epsilon)$ for obtaining an accuracy of $\epsilon$ with confidence $1 - \delta$. Their algorithm, Pegasos, essentially performs stochastic (sub)gradient descent but projects the solution back onto the $L_2$ ball of radius $1/\sqrt{\lambda}$. But, as our experiments show, performing an exact line search in the dual leads to a faster decrease in the value of primal objective. Note that Pegasos also can be used in an online setting. This, however, only applies whenever the empirical risk decomposes into individual loss terms (e.g. it is not applicable to multivariate performance scores).

The third related strand of research considers gradient descent in the primal with a line search to choose the optimal step size, see e.g. [2, Section 9.3.1]. Under assumptions of smoothness and strong convexity – that is, the objective function can be upper and lower bounded by quadratic functions – it can be shown that gradient descent with line search will converge to an accuracy of $\epsilon$ in $O(\log(1/\epsilon))$ steps. The problem here is the line search in the primal, since evaluating the regularized risk functional might be as expensive as computing its gradient, thus rendering a line search in the primal unattractive. On the other hand, the dual objective is relatively simple to evaluate, thus making the line search in the dual, as performed by our algorithm, computationally feasible.

Finally, we would like to point out connections to subgradient methods [7]. These algorithms are designed for nonsmooth functions, and essentially choose an arbitrary element of the subgradient set to perform a gradient descent like update. Let $\|J_w(w)\| \leq G$, and $B(w^*, r)$ denote a ball of radius $r$ centered around the minimizer of $J(w)$. By applying the analysis of Nedich and Bertsekas [7] to the regularized risk minimization problem with $\Omega(w) := \frac{\lambda}{2}\|w\|^2$, Ratliff et al. [9] showed that subgradient descent with a fixed, but sufficiently small, stepsize will converge linearly to $B(w^*, G/\lambda)$.

### References

[1] Y. Altun, A. J. Smola, and T. Hofmann. Exponential families for conditional random fields. In *UAI*, pages 2–9, 2004.

[2] S. Boyd and L. Vandenberghe. *Convex Optimization*. Cambridge University Press, 2004.

[3] J. Hiriart-Urruty and C. Lemaréchal. *Convex Analysis and Minimization Algorithms*. 1993.

[4] T. Joachims. A support vector method for multivariate performance measures. In *ICML*, pages 377–384, 2005.

[5] T. Joachims. Training linear SVMs in linear time. In *KDD*, 2006.

[6] S.-I. Lee, V. Ganapathi, and D. Koller. Efficient structure learning of Markov networks using L1-regularization. In *NIPS*, pages 817–824, 2007.

[7] A. Nedich and D. P. Bertsekas. Convergence rate of incremental subgradient algorithms. In *Stochastic Optimization: Algorithms and Applications*, pages 263–304. 2000.

[8] J. Nocedal and S. J. Wright. *Numerical Optimization*. Springer, 1999.

[9] N. Ratliff, J. Bagnell, and M. Zinkevich. (online) subgradient methods for structured prediction. In *Proc. of AIStats*, 2007.

[10] R. T. Rockafellar. *Convex Analysis*. Princeton University Press, 1970.

[11] S. Shalev-Shwartz and Y. Singer. Online learning meets optimization in the dual. In *COLT*, 2006.

[12] S. Shalev-Shwartz, Y. Singer, and N. Srebro. Pegasos: Primal estimated sub-gradient solver for SVM. In *ICML*, 2007.

[13] A. J. Smola, S. V. N. Vishwanathan, and Q. V. Le. Bundle methods for machine learning. *JMLR*, 2008. in preparation.

[14] B. Taskar, C. Guestrin, and D. Koller. Max-margin Markov networks. In *NIPS*, pages 25–32, 2004.

[15] C. H. Teo, Q. Le, A. Smola, and S. V. N. Vishwanathan. A scalable modular convex solver for regularized risk minimization. In *KDD*, 2007.

[16] I. Tsochantaridis, T. Joachims, T. Hofmann, and Y. Altun. Large margin methods for structured and interdependent output variables. *JMLR*, 6:1453–1484, 2005.

[17] V. Vapnik. *The Nature of Statistical Learning Theory*. Springer, 1995.

[18] T. Zhang. Sequential greedy approximation for certain convex optimization problems. *IEEE Trans. Information Theory*, 49(3):682–691, 2003.

